# Linear Classification and Selective Sampling Under Low Noise Conditions

**Giovanni Cavallanti**
DSI, Università degli Studi di Milano, Italy
cavallanti@dsi.unimi.it

**Nicolò Cesa-Bianchi**
DSI, Università degli Studi di Milano, Italy
cesa-bianchi@dsi.unimi.it

**Claudio Gentile**
DICOM, Università dell'Insubria, Italy
claudio.gentile@uninsubria.it

## Abstract

We provide a new analysis of an efficient margin-based algorithm for selective sampling in classification problems. Using the so-called Tsybakov low noise condition to parametrize the instance distribution, we show bounds on the convergence rate to the Bayes risk of both the fully supervised and the selective sampling versions of the basic algorithm. Our analysis reveals that, excluding logarithmic factors, the average risk of the selective sampler converges to the Bayes risk at rate $N^{-(1+\alpha)(2+\alpha)/2(3+\alpha)}$ where $N$ denotes the number of queried labels, and $\alpha > 0$ is the exponent in the low noise condition. For all $\alpha > \sqrt{3} - 1 \approx 0.73$ this convergence rate is asymptotically faster than the rate $N^{-(1+\alpha)/(2+\alpha)}$ achieved by the fully supervised version of the same classifier, which queries all labels, and for $\alpha \to \infty$ the two rates exhibit an exponential gap. Experiments on textual data reveal that simple variants of the proposed selective sampler perform much better than popular and similarly efficient competitors.

## 1 Introduction

In the standard online learning protocol for binary classification the learner receives a sequence of instances generated by an unknown source. Each time a new instance is received the learner predicts its binary label, and is then given the true label of the current instance before the next instance is observed. This protocol is natural in many applications, for instance weather forecasting or stock market prediction, because Nature (or the market) is spontaneously disclosing the true label after each learner's guess. On the other hand, in many other applications obtaining labels may be an expensive process. In order to address this problem, a variant of online learning that has been proposed is selective sampling. In this modified protocol the true label of the current instance is never revealed unless the learner decides to issue an explicit query. The learner's performance is then measured with respect to both the number of mistakes (made on the entire sequence of instances) and the number of queries. A natural sampling strategy is one that tries to identify labels which are likely to be useful to the algorithm, and then queries those ones only. This strategy somehow needs to combine a measure of utility of examples with a measure of confidence. In the case of learning with linear functions, a statistic that has often been used to quantify both utility and confidence is the margin. In [10] this approach was employed to define a selective sampling rule that queries a new label whenever the margin of the current instance, with respect to the current linear hypothesis, is smaller (in magnitude) than an adaptively adjusted threshold. Margins were computed using a linear learning algorithm based on an incremental version of Regularized linear Least-Squares (RLS) for classification. Although this selective sampling algorithm is efficient, and has simple variants working quite well in practice, the rate of convergence to the Bayes risk was never assessed in terms of natural distributional parameters, thus preventing a full understanding of the properties of this algorithm.

We improve on those results in several ways making three main contributions: (i) By coupling the Tsybakov low noise condition, used to parametrize the instance distribution, with the linear model of [10], defining the conditional distribution of labels, we prove that the fully supervised RLS (all labels are queried) converges to the Bayes risk at rate $\widetilde{\mathcal{O}}\big(n^{-(1+\alpha)/(2+\alpha)}\big)$ where $\alpha \geq 0$ is the noise exponent in the low noise condition. (ii) Under the same low noise condition, we prove that the RLS-based selective sampling rule of [10] converges to the Bayes risk at rate $\widetilde{\mathcal{O}}\big(n^{-(1+\alpha)/(3+\alpha)}\big)$, with labels being queried at rate $\widetilde{\mathcal{O}}\big(n^{-\alpha/(2+\alpha)}\big)$. Moreover, we show that similar results can be established for a mistake-driven (i.e., space and time efficient) variant. (iii) We perform experiments on a real-world medium-size dataset showing that variants of our mistake-driven sampler compare favorably with other selective samplers proposed in the literature, like the ones in [11, 16, 20].

**Related work.** Selective sampling, originally introduced by Cohn, Atlas and Ladner in [13, 14], differs from the active learning framework as in the latter the learner has more freedom in selecting which instances to query. For example, in Angluin's adversarial learning with queries (see [1] for a survey), the goal is to identify an unknown boolean function $f$ from a given class, and the learner can query the labels (i.e., values of $f$) of arbitrary boolean instances. Castro and Nowak [9] study a framework in which the learner also queries arbitrary domain points. However, in their case labels are stochastically related to instances (which are real vectors). They prove risk bounds in terms of nonparametric characterizations of both the regularity of the Bayes decision boundary and the behavior of the noise rate in its proximity. In fact, a large statistical literature on adaptive sampling and sequential hypothesis testing exists (see for instance the detailed description in [9]) which is concerned with problems that share similarities with active learning. The idea of querying small margin instances when learning linear classifiers has been explored several times in different active learning contexts. Campbell, Cristianini and Smola [8], and also Tong and Koller [23], study a pool-based model of active learning, where the algorithm is allowed to interactively choose which labels to obtain from an i.i.d. pool of unlabeled instances. A landmark result in the selective sampling protocol is the query-by-committee algorithm of Freund, Seung, Shamir and Tishby [17]. In the realizable (noise-free) case, and under strong distributional assumptions, this algorithm is shown to require exponentially fewer labels than instances when learning linear classifiers (see also [18] for a more practical implementation). An exponential advantage in the realizable case is also obtained with a simple variant of the Perceptron algorithm by Dasgupta, Kalai and Monteleoni [16], under the sole assumption that instances are drawn from the uniform distribution over the unit ball in $\mathbb{R}^d$. In the general statistical learning case, under no assumptions on the joint distribution of label and instances, selective sampling bears no such exponential advantage. For instance, Kääriäinen shows that, in order to approach the risk of the best linear classifier $f^*$ within error $\varepsilon$, at least $\Omega((\eta/\varepsilon)^2)$ labels are needed, where $\eta$ is the risk of $f^*$. A much more general nonparametric lower bound for active learning is obtained by Castro and Nowak [9]. General selective sampling strategies for the nonrealizable case have been proposed in [3, 4, 15]. However, none of these learning algorithms seems to be computationally efficient when learning linear classifiers in the general agnostic case.

## 2 Learning protocol and data model

We consider the following online selective sampling protocol. At each step $t = 1, 2, \ldots$ the sampling algorithm (or *selective sampler*) receives an instance $\boldsymbol{x}_t \in \mathbb{R}^d$ and outputs a binary prediction for the associated label $y_t \in \{-1, +1\}$. After each prediction, the algorithm has the option of "sampling" (issuing a query) in order to receive the label $y_t$. We call the pair $(\boldsymbol{x}_t, y_t)$ an example. After seeing the label $y_t$, the algorithm can choose whether or not to update its internal state using the new information encoded by $(\boldsymbol{x}_t, y_t)$.

We assume instances $\boldsymbol{x}_t$ are realizations of i.i.d. random variables $\boldsymbol{X}_t$ drawn from an unknown distribution on the surface of the unit Euclidean sphere in $\mathbb{R}^d$, so that $\|\boldsymbol{X}_t\| = 1$ for all $t \geq 1$. Following [10], we assume that labels $y_t$ are generated according to the following simple linear noise model: there exists a fixed and unknown vector $\boldsymbol{u} \in \mathbb{R}^d$, with Euclidean norm $\|\boldsymbol{u}\| = 1$, such that $\mathbb{E}\big[Y_t \,\big|\, \boldsymbol{X}_t = \boldsymbol{x}_t\big] = \boldsymbol{u}^\top \boldsymbol{x}_t$ for all $t \geq 1$. Hence $\boldsymbol{X}_t = \boldsymbol{x}_t$ has label 1 with probability $(1 + \boldsymbol{u}^\top \boldsymbol{x}_t)/2 \in [0, 1]$. Note that $\mathrm{SGN}(f^*)$, for $f^*(\boldsymbol{x}) = \boldsymbol{u}^\top \boldsymbol{x}$, is the Bayes optimal classifier for this noise model. In the following, all probabilities $\mathbb{P}$ and expectations $\mathbb{E}$ are understood with respect to the joint distribution of the i.i.d. data process $\{(\boldsymbol{X}_1, Y_1), (\boldsymbol{X}_2, Y_2), \ldots\}$. We use $\mathbb{P}_t$ to denote conditioning on $(\boldsymbol{X}_1, Y_1), \ldots, (\boldsymbol{X}_t, Y_t)$. Let $f : \mathbb{R}^d \to \mathbb{R}$ be an arbitrary measurable function. The *instantaneous regret* $R(f)$ is the excess risk of $\mathrm{SGN}(f)$ w.r.t. the Bayes risk, i.e., $R(f) = \mathbb{P}(Y_1 f(\boldsymbol{X}_1) < 0) - \mathbb{P}(Y_1 f^*(\boldsymbol{X}_1) < 0)$. Let $f_1, f_2, \ldots$ be a sequence of real functions

where each $f_t$ is measurable w.r.t. the $\sigma$-algebra generated by $(\boldsymbol{X}_1, Y_1), \ldots, (\boldsymbol{X}_{t-1}, Y_{t-1}), \boldsymbol{X}_t$. When $(\boldsymbol{X}_1, Y_1), \ldots, (\boldsymbol{X}_{t-1}, Y_{t-1})$ is understood from the context, we write $f_t$ as a function of $\boldsymbol{X}_t$ only. Let $R_{t-1}(f_t)$ be the instantaneous conditional regret $R_{t-1}(f_t) = \mathbb{P}_{t-1}(Y_t f_t(\boldsymbol{X}_t) < 0) - \mathbb{P}_{t-1}(Y_t f^*(\boldsymbol{X}_t) < 0)$. Our goal is to bound the expected cumulative regret $\mathbb{E}\big[R_0(f_1) + R_1(f_2) + \cdots + R_{n-1}(f_n)\big]$, as a function of $n$, and other relevant quantities. Observe that, although the learner's predictions can only depend on the queried examples, the regret is computed over *all* time steps, including the ones when the selective sampler did not issue a query. In order to model the distribution of the instances around the hyperplane $\boldsymbol{u}^\top \boldsymbol{x} = 0$, we use Mammen-Tsybakov *low noise condition* [24]:

$$\text{There exist } c > 0 \text{ and } \alpha \geq 0 \text{ such that} \quad \mathbb{P}\big(|f^*(\boldsymbol{X}_1)| < \varepsilon\big) \leq c\,\varepsilon^\alpha \quad \text{for all } \varepsilon > 0. \tag{1}$$

When the noise exponent $\alpha$ is 0 the low noise condition becomes vacuous. In order to study the case $\alpha \to \infty$, one can use the following equivalent formulation of (1) —see, e.g., [5], $\mathbb{P}\big(f^*(\boldsymbol{X}_1)f(\boldsymbol{X}_1) < 0\big) \leq c\,R(f)^{\alpha/(1+\alpha)}$ for all measurable $f : \mathbb{R}^d \to \mathbb{R}$. With this formulation, one can show that $\alpha \to \infty$ implies the *hard margin condition* $|f^*(\boldsymbol{X}_1)| \geq 1/(2c)$ w.p. 1.

## 3 Algorithms and theoretical analysis

We consider linear classifiers predicting the value of $Y_t$ through $\textsc{sgn}(\boldsymbol{w}_t^\top \boldsymbol{X}_t)$, where $\boldsymbol{w}_t \in \mathbb{R}^d$ is a dynamically updated weight vector which might be intended as the current estimate for $\boldsymbol{u}$. Our $\boldsymbol{w}_t$ is an RLS estimator defined over the set of previously queried examples. More precisely, let $N_t$ be the number of queried examples during the first $t$ time steps, $S_{t-1} = \big[\boldsymbol{x}_1', \ldots, \boldsymbol{x}_{N_{t-1}}'\big]$ be the matrix of the queried instances up to time $t-1$, and $\boldsymbol{y}_{t-1} = \big[y_1', \ldots, y_{N_{t-1}}'\big]^\top$ be the vector of the corresponding labels. Then the RLS estimator is defined by

$$\boldsymbol{w}_t = \big(I + S_{t-1} S_{t-1}^\top + \boldsymbol{x}_t \boldsymbol{x}_t^\top\big)^{-1} S_{t-1}\, \boldsymbol{y}_{t-1}\,, \tag{2}$$

where $I$ is the $d \times d$ identity matrix. Note that $\boldsymbol{w}_t$ depends on the current instance $\boldsymbol{x}_t$. The RLS estimator in this particular form has been first considered by Vovk [25] and by Azoury and Warmuth [2]. Compared to standard RLS, here $\boldsymbol{x}_t$ acts by futher reducing the variance of $\boldsymbol{w}_t$. We use $\widehat{\Delta}_t$ to denote the margin $\boldsymbol{w}_t^\top \boldsymbol{X}_t$ whenever $\boldsymbol{w}_t$ is understood from the context. Thus $\widehat{\Delta}_t$ is the current approximation to $\Delta_t$. Note that $\widehat{\Delta}_t$ is measurable w.r.t. the $\sigma$-algebra generated by $(\boldsymbol{X}_1, Y_1), \ldots, (\boldsymbol{X}_{t-1}, Y_{t-1}), \boldsymbol{X}_t$. We also use $\Delta_t$ to denote the Bayes margin $f^*(\boldsymbol{X}_t) = \boldsymbol{u}^\top \boldsymbol{X}_t$.

The RLS estimator (2) can be stored in space $\Theta(d^2)$, which we need for the inverse of $I + S_{t-1} S_{t-1}^\top + \boldsymbol{x}_t \boldsymbol{x}_t^\top$. Moreover, using a standard formula for small-rank adjustments of inverse matrices, we can compute updates and predictions in time $\Theta(d^2)$. The algorithm in (2) can also be expressed in dual variable form. This is needed, for instance, when we want to use the feature expansion facility provided by kernel functions. In this case, at time $t$ the RLS estimator (2) can be represented in $O(N_{t-1}^2)$ space. The update time is also quadratic in $N_{t-1}$.

Our first result establishes a regret bound for the fully supervised algorithm, i.e., the algorithm that predicts using RLS as in (2), queries the label of *every* instance, and stores all examples. This result is the baseline against which we measure the performance of our selective sampling algorithm. The regret bound is expressed i.t.o. the whole spectrum of the process covariance matrix $\mathbb{E}[\boldsymbol{X}_1 \boldsymbol{X}_1^\top]$.

**Theorem 1** *Assume the low noise condition (1) holds with exponent $\alpha \geq 0$ and constant $c > 0$. Then the expected cumulative regret after $n$ steps of the fully supervised algorithm based on (2) is bounded by* $\mathbb{E}\left[\big(4c(1 + \ln|I + S_n S_n^\top|)\big)^{\frac{1+\alpha}{2+\alpha}}\right] n^{\frac{1}{2+\alpha}}$. *This, in turn, is bounded from above by*

$$\left(4c\Big(1 + \sum_{i=1}^d \ln(1 + n\lambda_i)\Big)\right)^{\frac{1+\alpha}{2+\alpha}} n^{\frac{1}{2+\alpha}} = \mathcal{O}\left(\big(d\ln n\big)^{\frac{1+\alpha}{2+\alpha}} n^{\frac{1}{2+\alpha}}\right). \textit{ Here } |\cdot| \textit{ denotes the determinant of a matrix, } S_n = \big[\boldsymbol{X}_1, \boldsymbol{X}_2, \ldots, \boldsymbol{X}_n\big], \textit{ and } \lambda_i \textit{ is the } i\text{-th eigenvalue of } \mathbb{E}[\boldsymbol{X}_1 \boldsymbol{X}_1^\top].$$

When $\alpha = 0$ (corresponding to a vacuous noise condition) the bound of Theorem 1 reduces to $\mathcal{O}\big(\sqrt{d\,n\ln n}\big)$. When $\alpha \to \infty$ (corresponding to a hard margin condition) the bound gives the logarithmic behavior $\mathcal{O}\big(d\ln n\big)$. Notice that $\sum_{i=1}^d \ln(1 + n\lambda_i)$ is substantially smaller than $d\ln n$ whenever the spectrum of $\mathbb{E}[\boldsymbol{X}_1 \boldsymbol{X}_1^\top]$ is rapidly decreasing. In fact, the second bound is clearly meaningful even when $d = \infty$, while the third one only applies to the finite dimensional case.

**Parameters:** $\lambda > 0$, $\rho_t > 0$ for each $t \geq 1$.
**Initialization:** weight vector $\boldsymbol{w} = (0, \ldots, 0)^\top$; storage counter $N = 0$.
At each time $t = 1, 2, \ldots$ do the following:

1. Observe instance $\boldsymbol{x}_t \in \mathbb{R}^d$: $||\boldsymbol{x}_t|| = 1$;

2. Predict the label $y_t \in \{-1, 1\}$ with $\text{SGN}(\boldsymbol{w}_t^\top \boldsymbol{x}_t)$, where $\boldsymbol{w}_t$ is as in (2).

3. If $N \leq \rho_t$ then query label $y_t$ and store $(\boldsymbol{x}_t, y_t)$;

4. Else if $\widehat{\Delta}_t^2 \leq \frac{128 \ln t}{\lambda N}$ then schedule the query of $y_{t+1}$;

5. If $(\boldsymbol{x}_t, y_t)$ is scheduled to be stored, then increment $N$ and update $\boldsymbol{w}_t$ using $(\boldsymbol{x}_{t+1}, y_{t+1})$.

<div align="center">Figure 1: The selective sampling algorithm.</div>

Fast rates of convergence have typically been proven for batch-style algorithms, such as empirical risk minimizers and SVM (see, e.g., [24, 22]), rather than for online algorithms. A reference closer to our paper is Ying and Zhou [26], where the authors prove bounds for online linear classification using the low noise condition (1), though under different distributional assumptions.

Our second result establishes a new regret bound, under low noise conditions, for the selective sampler introduced in [10]. This variant, described in Figure 1, queries all labels (and stores all examples) during an initial stage of length at least $(16d)/\lambda^2$, where $\lambda$ denotes the smallest nonzero eigenvalue of the process covariance matrix $\mathbb{E}[\boldsymbol{X}_1 \boldsymbol{X}_1^\top]$. When this transient regime is over, the sampler issues a query at time $t$ based on both the query counter $N_{t-1}$ and the margin $\widehat{\Delta}_t$. Specifically, if evidence is collected that the number $N_{t-1}$ of stored examples is smaller than our current estimate of $1/\widehat{\Delta}_t^2$, that is if $\widehat{\Delta}_t^2 \leq (128 \ln t)/(\lambda N_{t-1})$, then we query (and store) the label of the *next* instance $\boldsymbol{x}_{t+1}$. Note that the margin threshold explicitly depends, through $\lambda$, on additional information about the data-generating process. This additional information is needed because, unlike the fully supervised classifier of Theorem 1, the selective sampler queries labels at random steps. This prevents us from bounding the sum of conditional variances of the involved RLS estimator through $\ln |I + S_n S_n^\top|$, as we can do when proving Theorem 1 (see below). Instead, we have to individually bound each conditional variance term via the smallest empirical eigenvalue of the correlation matrix. The transient regime in Figure 1 is exactly needed to ensure that this smallest empirical eigenvalue gets close enough to $\lambda$. Compared to the analysis contained in [10], we are able to better capture the two main aspects of the selective sampling protocol: First, we control the probability of making a mistake when we do not query labels; second, the algorithm is able to adaptively optimize the sampling rate by exploiting the additional information provided by the examples having small margin. The appropriate sampling rate clearly depends on the (unknown) amount of noise $\alpha$ which the algorithm implicitly learns on the fly. In this respect, our algorithm is more properly an *adaptive* sampler, rather than a selective sampler. Finally, we stress that it is fairly straightforward to add to the algorithm in Figure 1 a mistake-driven rule for storing examples. Such a rule provides that, when a small margin is detected, a query be issued (and the next example be stored) only if $\text{SGN}(\widehat{\Delta}_t) \neq y_t$ (i.e., only if the current prediction is mistaken). This turns out to be highly advantageous from a computational standpoint, because of the sparsity of the computed solution. It is easy to adapt our analysis to obtain even for this algorithm the same regret bound as the one established in Theorem 2. However, in this case we can only give guarantees on the expected number of stored examples (which can indeed be much smaller than the actual number of queried labels).

**Theorem 2** *Assume the low noise condition (1) holds with unknown exponent $\alpha \geq 0$ and assume the selective sampler of Figure 1 is run with $\rho_t = \frac{16}{\lambda^2} \max\{d, \ln t\}$. Then, after $n$ steps, the expected cumulative regret is bounded by* $O\left( \frac{d + \ln n}{\lambda^2} + \left( \frac{\ln n}{\lambda} \right)^{\frac{1+\alpha}{3+\alpha}} n^{\frac{2}{3+\alpha}} \right)$ *whereas the expected number of queried labels (including the stored ones) is bounded by* $O\left( \frac{d + \ln n}{\lambda^2} + \left( \frac{\ln n}{\lambda} \right)^{\frac{\alpha}{2+\alpha}} n^{\frac{2}{2+\alpha}} \right)$ .

The proof, sketched below, hinges on showing that $\widehat{\Delta}_t$ is an almost unbiased estimate of the true margin $\Delta_t$, and relies on known concentration properties of i.i.d. processes. In particular, we show that our selective sampler is able to adaptively estimate the number of queries needed to ensure a $1/t$ increase of the regret when a query is not issued at time $t$.

As expected, when we compare our semi-supervised selective sampler (Theorem 2) to the fully supervised "yardstick" (Theorem 1), we see that the per-step regret of the former vanishes at a significantly slower rate than the latter, i.e., $n^{-\frac{1+\alpha}{3+\alpha}}$ vs. $n^{-\frac{1+\alpha}{2+\alpha}}$. Note, however, that the per-step regret of the semi-supervised algorithm vanishes faster than its fully-supervised counterpart when both regrets are expressed in terms of the number $N$ of issued queries. To see this consider first the case $\alpha \to \infty$ (the hard margin case, essentially analyzed in [10]). Then both algorithms have a per-step regret of order $(\ln n)/n$. However, since the semi-supervised algorithm makes only $N = O(\ln n)$ queries, we have that, as a function of $N$, the per-step regret of the semi-supervised algorithm is of order $N/e^N$ where the fully supervised has only $(\ln N)/N$. We have thus recovered the exponential advantage observed in previous works [16, 17]. When $\alpha = 0$ (vacuous noise conditions), the per-step regret rates in terms of $N$ become (excluding logarithmic factors) of order $N^{-1/3}$ in the semi-supervised case and of order $N^{-1/2}$ in the fully supervised case. Hence, there is a critical value of $\alpha$ where the semi-supervised bound becomes better. In order to find this critical value we write the rates of the per-step regret for $0 \le \alpha < \infty$ obtaining $N^{-\frac{(1+\alpha)(2+\alpha)}{2(3+\alpha)}}$ (semi-supervised algorithm) and $N^{-\frac{1+\alpha}{2+\alpha}}$ (fully supervised algorithm). By comparing the two exponents we find that, asymptotically, the semi-supervised rate is better than the fully supervised one for all values of $\alpha > \sqrt{3} - 1$. This indicates that selective sampling is advantageous when the noise level (as modeled by the Mammen-Tsybakov condition) is not too high. Finally, observe that the way it is stated now, the bound of Theorem 2 only applies to the finite-dimensional ($d < \infty$) case. It turns out this is a fixable artifact of our analysis, rather than an intrinsic limitation of the selective sampling scheme in Figure 1. See Remark 3 below.

**Proof of Theorem 1.** The proof proceeds by relating the classification regret to the square loss regret via a comparison theorem. The square loss regret is then controlled by applying a known pointwise bound. For all measurable $f : \mathbb{R}^d \to \mathbb{R}$, let $R_\phi(f) = \mathbb{E}\big[\big(1 - Y_1\, f(\boldsymbol{X}_1)\big)^2 - \big(1 - Y_1\, f^*(\boldsymbol{X}_1)^2\big)\big]$ be the square loss regret, and $R_{t-1,\phi}$ its conditional version. We apply the comparison theorem from [5] with the $\psi$-transform function $\psi(z) = z^2$ associated with the square loss. Under the low noise condition (1) this yields $R(f) \le \big(4c\, R_\phi(f)\big)^{\frac{1+\alpha}{2+\alpha}}$ for all measurable $f$. We thus have $\mathbb{E}\big[\sum_{t=1}^n R_{t-1}(f_t)\big] \le \mathbb{E}\Big[\sum_{t=1}^n \big(4c\, R_{\phi,t-1}(f_t)\big)^{\frac{1+\alpha}{2+\alpha}}\Big] \le \mathbb{E}\Big[n\big(\frac{4c}{n} \sum_{t=1}^n R_{\phi,t-1}(f_t)\big)^{\frac{1+\alpha}{2+\alpha}}\Big]$, the last term following from Jensen's inequality. Further, we observe that in our probabilistic model $f^*(\boldsymbol{x}) = \boldsymbol{u}^\top \boldsymbol{x}$ is Bayes optimal for the square loss. In fact, for any unit norm $\boldsymbol{x} \in \mathbb{R}^d$, we have $f^*(\boldsymbol{x}) = \operatorname{arginf}_{z \in \mathbb{R}}\Big((1 - z)^2 \frac{1+\boldsymbol{u}^\top \boldsymbol{x}}{2} + (1 + z)^2 \frac{1-\boldsymbol{u}^\top \boldsymbol{x}}{2}\Big) = \boldsymbol{u}^\top \boldsymbol{x}$. Hence $\sum_{t=1}^n R_{\phi,t-1}(f_t) = \sum_{t=1}^n \big((Y_t - \boldsymbol{w}_t^\top \boldsymbol{X}_t)^2 - (Y_t - \boldsymbol{u}^\top \boldsymbol{X}_t)^2\big)$ which, in turn, can be bounded pointwise (see, e.g., [12, Theorem 11.8]) by $1 + \ln\big|I + S_n S_n^\top\big|$. Putting together gives the first bound. Next, we take the bound just obtained and apply Jensen's inequality twice, first to the concave function $(\cdot)^{\frac{1+\alpha}{2+\alpha}}$ of a real argument, and then to the concave function $\ln|\cdot|$ of a (positive definite) matrix argument. Observing that $\mathbb{E}S_n S_n^\top = \mathbb{E}[\sum_{t=1}^n \boldsymbol{X}_t \boldsymbol{X}_t^\top] = n\, \mathbb{E}\boldsymbol{X}_1 \boldsymbol{X}_1^\top$ yields the second bound. The third bound derives from the second one just by using $\lambda_i \le 1$. $\qquad\square$

**Proof sketch of Theorem 2.** We aim at bounding from above the cumulative regret $\sum_{t=1}^n \Big(\mathbb{P}(Y_t\, \widehat{\Delta}_t < 0) - \mathbb{P}(Y_t\, \Delta_t < 0)\Big)$ which, according to our probabilistic model, can be shown to be at most $c\, n\, \varepsilon^{1+\alpha} + \sum_{t=1}^n \mathbb{P}(\Delta_t\, \widehat{\Delta}_t \le 0,\ |\Delta_t| \ge \varepsilon)$. The last sum is upper bounded by

$$\underbrace{\sum_{t=1}^n \mathbb{P}\left(N_{t-1} \le \rho_t\right)}_{(\text{I})} + \underbrace{\sum_{t=1}^n \mathbb{P}\left(\widehat{\Delta}_t^2 \le \frac{128 \ln t}{\lambda N_{t-1}},\ N_{t-1} > \rho_t,\ |\Delta_t| \ge \varepsilon\right)}_{(\text{II})}$$

$$+ \underbrace{\sum_{t=1}^n \mathbb{P}\left(\Delta_t\, \widehat{\Delta}_t \le 0,\ \widehat{\Delta}_t^2 > \frac{128 \ln t}{\lambda N_{t-1}},\ N_{t-1} > \rho_t\right)}_{(\text{III})}.$$

where: (I) are the initial time steps; (II) are the time steps on which we trigger the query of the next label (because $\widehat{\Delta}_t^2$ is smaller than the threshold at time $t$); (III) are the steps that do not trigger any queries at all.

Note that (III) bounds the regret over non-sampled examples. In what follows, we sketch the way we bound each of the three terms separately. A bound on (I) is easily obtained as (I) $\leq \rho_n = \mathcal{O}(\frac{d+\ln n}{\lambda^2})$ just because $\rho_n \geq \rho_t$ for all $t \leq n$. To bound (II) and (III) we need to exploit the fact that the subsequence of stored instances and labels is a sequence of i.i.d. random variables distributed as $(\boldsymbol{X}_1, Y_1)$, see [10]. This allows us to carry out a (somewhat involved) bias-variance analysis showing that for any fixed number $N_{t-1} = s$ of stored examples, $\widehat{\Delta}_t$ is an almost unbiased estimator of $\Delta_t$, whose bias and variance tend to vanish as $1/s$ when $s$ is sufficiently large. In particular, if $|\Delta_t| \geq \varepsilon$ then $\widehat{\Delta}_t \approx \Delta_t$ as long as $N_{t-1}$ is of the order of $\frac{\ln n}{\lambda \varepsilon^2}$. The variance of $\widehat{\Delta}_t$ is controlled by known results (the one we used is [21, Theorem 4.2]) on the concentration of eigenvalues of an empirical correlation matrix $\frac{1}{s}\sum_i \boldsymbol{X}_i \boldsymbol{X}_i^\top$ to the eigenvalues of the process covariance matrix $\mathbb{E}[\boldsymbol{X}_1 \boldsymbol{X}_1^\top]$. For such a result to apply, we have to impose that $N_{t-1} \geq \rho_t$. By suitably combining these concentration results we can bound term (II) by $\mathcal{O}(\frac{d+\ln n}{\lambda^2} + \frac{\ln n}{\lambda \varepsilon^2})$ and term (III) by $\mathcal{O}(\ln n)$.

Putting together and choosing $\varepsilon$ of the order of $\left(\frac{\ln n}{\lambda n}\right)^{\frac{1+\alpha}{3+\alpha}}$ gives the desired regret bound. The bound on the number of queried labels is obtained in a similar way. $\qquad\square$

**Remark 3** *The linear dependence on $d$ in Theorem 2 derives from a direct application of the concentration results in [21]. In fact, it is possible to take into account in a fairly precise manner the way the process spectrum decreases (e.g., [6, 7]), thereby extending the above analysis to the infinite-dimensional case. In this paper, however, we decided to stick to the simpler analysis leading to Theorem 2, since the resulting bounds would be harder to read, and would somehow obscure understanding of regret and sampling rate behavior as a function of $n$.*

## 4 Experimental analysis

In evaluating the empirical performance of our selective sampling algorithm, we consider two additional variants obtained by slightly modifying Step 4 in Figure 1. The first variant (which we just call SS, Selective Sampler) queries the current label instead of the next one. The rationale here is that we want to leverage the more informative content of small margin instances. The second variant is a mistake-driven version (referred to as SSMD, Selective Sampling Mistake Driven) that queries the current label (and stores the corresponding example) only if the label gets mispredicted. For clarity, the algorithm in Figure 1 will then be called SSNL (Selective Sampling Next Label) since it queries the next label whenever a small margin is observed. For all three algorithms we dropped the intial transient regime (Step 3 in Figure 1).

We run our experiments on the first, in chronological order, 40,000 newswire stories from the Reuters Corpus Volume 1 dataset (RCV1). Every example in this dataset is encoded as a vector of real attributes computed through a standard TF-IDF bag-of-words processing of the original news stories, and is tagged with zero or more labels from a set of 102 classes. The online categorization of excerpts from a newswire feed is a realistic learning problem for selective sampling algorithms since a newswire feed consists of a large amount of uncategorized data with a high labeling cost. The classification performance is measured using a macroaveraged $F$-measure $2RP/(R + P)$, where $P$ is the precision (fraction of correctly classified documents among all documents that were classified positive for the given topic) and $R$ is the recall (fraction of correctly classified documents among all documents that are labelled with the given topic). All algorithms presented here are evaluated using dual variable implementations and linear kernels.

The results are summarized in Figures 2 and 3. The former only refers to (an average over) the 50 most frequent categories, while the latter includes them all. In Figure 2 (left) we show how SSMD compares to SSNL, and to its most immediate counterpart, SS. In Figure 2 (right) we compare SSMD to other algorithms that are known to have good empirical performance, including the second-order version of the label efficient classifier (SOLE), as described in [11], and the DKMPERC variant of the DKM algorithm (see, e.g., [16, 20]). DKMPERC differs from DKM since it adopts a standard perceptron update rule. The perceptron algorithm (PERC) and its second-order counterpart (SOP) are reported here as a reference, since they are designed to query all labels. In particular, SOP is a mistake-driven variant of the algorithm analyzed in Theorem 1. It is reasonable to assume that in a selective sampling setup we are interested in the performance achieved when the fraction of queried labels stays below some threshold, say $10\%$. In this range of sampling rate, SSMD has the steepest increase in the achieved $F$-measure, and surpasses any other algorithm. Unsurprisingly, as the number of queried labels gets larger, SSMD, SOLE and SOP exhibit similar behaviors. Moreover, the less than ideal plot of SSNL seems to confirm the intuition that querying small margin instances

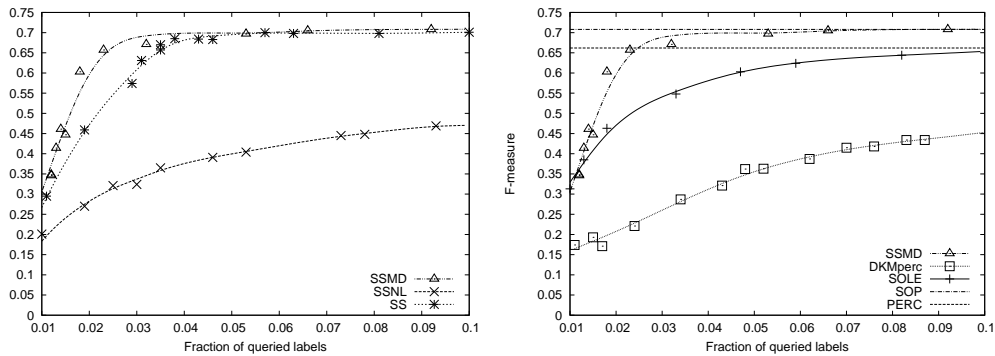

Figure 2: Average $F$-measure obtained by different algorithms after $40,000$ examples, as a function of the number of queried labels. The average only refers to the 50 most frequent categories. Points are obtained by repeatedly running each algorithm with different values of parameters (in Figure 1, the relevant parameter is $\lambda$). Trend lines are computed as approximate cubic splines connecting consecutive points.

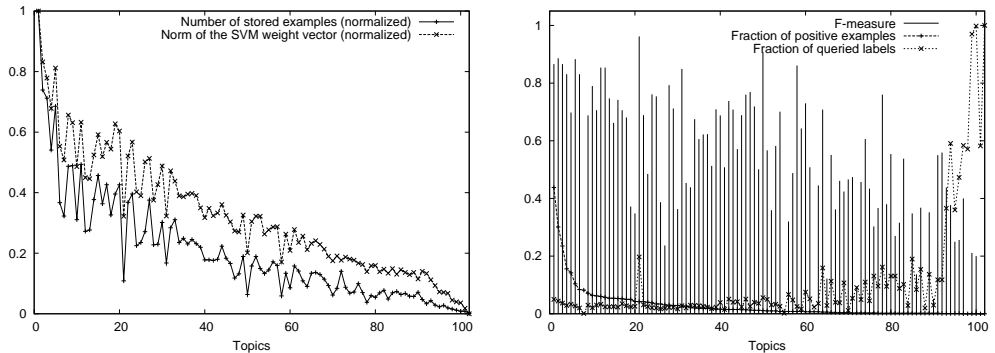

Figure 3: Left: Correlation between the fraction of stored examples and the difficulty of each binary task, as measured by the separation margin. Right: $F$-measure achieved on the different binary classification tasks compared to the number of positive examples in each topic, and to the fraction of queried labels (including the stored ones). In both plots, topics are sorted by decreasing frequency of positive examples. The two plots are produced by SSMD with a specific value of the $\lambda$ parameter. Varying $\lambda$ does not significantly alter the reported trend.

provides a significant advantage. Under our test conditions DKMPERC proved ineffective, probably because most tasks in the RCV1 dataset are not linearly separable. A similar behavior was observed in [20]. It is fair to remark that DKMPERC is a perceptron-like linear-threshold classifier while the other algorithms considered here are based on the more computationally intensive ridge regression-like procedure.

In our selective sampling framework it is important to investigate how harder problems influence the sampling rate of an algorithm and, for each binary problem, to assess the impact of the number of positive examples on F-measure performance. Coarsely speaking, we would expect that the hard topics are the infrequent ones. Here we focus on SSMD since it is reasonably the best candidate, among our selective samplers, as applied to real-world problems. In Figure 3 (left) we report the fraction of examples stored by SSMD on each of the 102 binary learning tasks (i.e., on each individual topic, including the infrequent ones), and the corresponding levels of $F$-measure and queried labels (right). Note that in both plots topics are sorted by frequency with the most frequent categories appearing on the left. We represent the difficulty of a learning task by the norm of the weight vector obtained by running the C-SVM algorithm on that task[1]. Figure 3 (left) clearly shows that SSMD rises the storage rate on difficult problems. In particular, even if two different tasks have largely different numbers of positive examples, the storage rate achieved by SSMD on those tasks may be

similar when the norm of the weight vectors computed by C-SVM is nearly the same. On the other hand, the right plot shows (to our surprise) that the achieved F-measure is fairly independent of the number of positive examples, but this independence is obtained at the cost of querying more and more labels. In other words, SSMD seems to realize the difficulty of learning infrequent topics and, in order to achieve a good F-measure performance, it compensates by querying many more labels.

## Footnotes

[1]The actual values were computed using SVM-LIGHT [19] with default parameters. Since the examples in the Reuters Corpus Volume 1 are cosine normalized, the choice of default parameters amounts to indirectly setting the parameter $C$ to approximately $1.0$.

## References

[1] D. Angluin. Queries revisited. In *12th ALT*, pages 12–31. Springer, 2001.

[2] K.S. Azoury and M.K. Warmuth. Relative loss bounds for on-line density estimation with the exponential family of distributions. *Machine Learning*, 43(3):211–246, 2001.

[3] M.F. Balcan, A. Beygelzimer, and J. Langford. Agnostic active learning. In *23rd ICML*, pages 65–72. ACM Press, 2006.

[4] M.F. Balcan, A. Broder, and T. Zhang. Margin-based active learning. In *20th COLT*, pages 35–50. Springer, 2007.

[5] P.L. Bartlett, M.I. Jordan, and J.D. McAuliffe. Convexity, classification, and risk bounds. *JASA*, 101(473):138–156, 2006.

[6] G. Blanchard, O. Bousquet, and L. Zwald. Statistical properties of kernel principal component analysis. *Machine Learning*, 66:259–294, 2007.

[7] M.L. Braun. Accurate error bounds for the eigenvalues of the kernel matrix. *JMLR*, 7:2303–2328, 2006.

[8] C. Campbell, N. Cristianini, and A. Smola. Query learning with large margin classifiers. In *17th ICML*, pages 111–118. Morgan Kaufmann, 2000.

[9] R. Castro and R.D. Nowak. Minimax bounds for active learning. *IEEE Trans. IT*, 2008. To appear.

[10] N. Cesa-Bianchi, A. Conconi, and C. Gentile. Learning probabilistic linear-threshold classifiers via selective sampling. In *16th COLT*, pages 373–387. Springer, 2003.

[11] N. Cesa-Bianchi, C. Gentile, and L. Zaniboni. Worst-case analysis of selective sampling for linear classification. *JMLR*, 7:1205–1230, 2006.

[12] N. Cesa-Bianchi and G. Lugosi. *Prediction, Learning, and Games*. Cambridge University Press, 2006.

[13] D. Cohn, L. Atlas, and R. Ladner. Improving generalization with active learning. *Machine Learning*, 15(2):201–221, 1994.

[14] R. Cohn, L. Atlas, and R. Ladner. Training connectionist networks with queries and selective sampling. In *NIPS 2*. MIT Press, 1990.

[15] S. Dasgupta, D. Hsu, and C. Monteleoni. A general agnostic active learning algorithm. In *NIPS 20*, pages 353–360. MIT Press, 2008.

[16] S. Dasgupta, A. T. Kalai, and C. Monteleoni. Analysis of Perceptron-based active learning. In *18th COLT*, pages 249–263. Springer, 2005.

[17] Y. Freund, S. Seung, E. Shamir, and N. Tishby. Selective sampling using the query by committee algorithm. *Machine Learning*, 28(2/3):133–168, 1997.

[18] R. Gilad-Bachrach, A. Navot, and N. Tishby. Query by committee made real. *NIPS*, 18, 2005.

[19] T. Joachims. Making large-scale SVM learning practical. In B. Schölkopf, C. Burges, and A. Smola, editors, *Advances in Kernel Methods: Support Vector Learning*. MIT Press, 1999.

[20] C. Monteleoni and M. Kääriäinen. Practical online active learning for classification. In *24th IEEE CVPR*, pages 249–263. IEEE Computer Society Press, 2007.

[21] J. Shawe-Taylor, C.K.I. Williams, N. Cristianini, and J. Kandola. On the eigenspectrum of the Gram matrix and the generalization error of kernel-PCA. *IEEE Trans. IT*, 51(7):2510–2522, 2005.

[22] I. Steinwart and C. Scovel Fast Rates for Support Vector Machines using Gaussian Kernels *Annals of Statistics*, 35: 575-607, 2007.

[23] S. Tong and D. Koller. Support vector machine active learning with applications to text classification. In *17th ICML*, pages 999–1006. Morgan Kaufmann, 2000.

[24] A. Tsybakov. Optimal aggregation of classifiers in statistical learning. *The Annals of Statistics*, 32(1):135–166, 2004.

[25] V. Vovk. Competitive on-line statistics. *International Statistical Review*, 69:213–248, 2001.

[26] Y. Ying and D.X. Zhou. Online regularized classification algorithms. *IEEE Transactions on Information Theory*, 52:4775–4788, 2006.
